# Convergence of The Wake-Sleep Algorithm

**Shiro Ikeda**
PRESTO, JST
Wako, Saitama, 351-0198, Japan
shiro@brain.riken.go.jp

**Shun-ichi Amari**
RIKEN Brain Science Institute
Wako, Saitama, 351-0198, Japan
amari@brain.riken.go.jp

**Hiroyuki Nakahara**
RIKEN Brain Science Institute
hiro@brain.riken.go.jp

## Abstract

The W-S (Wake-Sleep) algorithm is a simple learning rule for the models with hidden variables. It is shown that this algorithm can be applied to a factor analysis model which is a linear version of the Helmholtz machine. But even for a factor analysis model, the general convergence is not proved theoretically. In this article, we describe the geometrical understanding of the W-S algorithm in contrast with the EM (Expectation-Maximization) algorithm and the *em* algorithm. As the result, we prove the convergence of the W-S algorithm for the factor analysis model. We also show the condition for the convergence in general models.

## 1 INTRODUCTION

The W-S algorithm[5] is a simple Hebbian learning algorithm. Neal and Dayan applied the W-S algorithm to a factor analysis model[7]. This model can be seen as a linear version of the Helmholtz machine[3]. As it is mentioned in[7], the convergence of the W-S algorithm has not been proved theoretically even for this simple model.

From the similarity of the W-S and the EM algorithms and also from empirical results, the W-S algorithm seems to work for a factor analysis model. But there is an essential difference between the W-S and the EM algorithms. In this article, we show the *em* algorithm[2], which is the information geometrical version of the EM algorithm, and describe the essential difference. From the result, we show that we cannot rely on the similarity for the reason of the W-S algorithm to work. However, even with this difference, the W-S algorithm works on the factor analysis model and we can prove it theoretically. We show the proof and also show the condition of the W-S algorithm to work in general models.

## 2  FACTOR ANALYSIS MODEL AND THE W-S ALGORITHM

A factor analysis model with a single factor is defined as the following generative model,

**Generative model**                            $x = \mu + yg + \epsilon,$

where $x = (x_1, \cdots, x_n)^T$ is a $n$ dimensional real-valued visible inputs, $y \sim \mathcal{N}(0,1)$ is the single invisible factor, $g$ is a vector of "factor loadings", $\mu$ is the overall means vector which is set to be zero in this article, and $\epsilon \sim \mathcal{N}(0, \Sigma)$ is the noise with a diagonal covariance matrix, $\Sigma = \mathrm{diag}(\sigma_i^2)$. In a Helmholtz machine, this generative model is accompanied by a recognition model which is defined as,

**Recognition model**                           $y = r^T x + \delta,$

where $r$ is the vector of recognition weights and $\delta \sim \mathcal{N}(0, s^2)$ is the noise.

When data $x_1, \cdots, x_N$ is given, we want to estimate the MLE(Maximum Likelihood Estimator) of $g$ and $\Sigma$. The W-S algorithm can be applied[7] for learning of this model.

**Wake-phase:** From the training set $\{x_s\}$ choose a number of $x$ randomly and for each data, generate $y$ according to the recognition model $y = r_t^T x + \delta, \delta \sim \mathcal{N}(0, s_t^2)$. Update $g$ and $\Sigma$ as follows using these $x$'s and $y$'s, where $\alpha$ is a small positive number and $\beta$ is slightly less than 1.

$$g_{t+1} = g_t + \alpha \overline{(x - g_t y)y} \tag{1}$$

$$\sigma_{i,t+1}^2 = \beta \sigma_{i,t}^2 + (1 - \beta)\overline{(x_i - g_{i,t}y)^2}, \tag{2}$$

where $^-$ denotes the averaging over the chosen data.

**Sleep-phase:** According to the updated generative model $x = yg_{t+1} + \epsilon, y \sim \mathcal{N}(0,1), \epsilon \sim \mathcal{N}(0, \mathrm{diag}(\sigma_{t+1}^2))$, generate a number of $x$ and $y$. And update $r$ and $s^2$ as,

$$r_{t+1} = r_t + \alpha \overline{(y - r_t^T x)x} \tag{3}$$

$$s_{t+1}^2 = \beta s_t^2 + (1 - \beta)\overline{(y - r_t^T x)^2}. \tag{4}$$

By iterating these phases, they try to find the MLE as the converged point.

For the following discussion, let us define two probability densities $p$ and $q$, where $p$ is the density of the generative model, and $q$ is that of the recognition model.

Let $\theta = (g, \Sigma)$, and the generative model gives the density function of $x$ and $y$ as,

$$p(y, x; \theta) = \exp\left(-\frac{1}{2}(y\ x^T)A\begin{pmatrix} y \\ x \end{pmatrix} - \psi(\theta)\right) \tag{5}$$

$$A = \left(\begin{array}{c|c} 1 + g^T \Sigma^{-1} g & -g^T \Sigma^{-1} \\ \hline -\Sigma^{-1} g & \Sigma^{-1} \end{array}\right), \psi(\theta) = \frac{1}{2}\left(\sum \log \sigma_i^2 + (n+1)\log 2\pi\right),$$

while the recognition model gives the distribution of $y$ conditional to $x$ as the following,

$$q(y|x; \eta) \sim \mathcal{N}(r^T x, s^2),$$

where, $\eta = (r, s^2)$. From the data $x_1, \cdots, x_N$, we define,

$$C = \frac{1}{N}\sum_{s=1}^{N} x_s x_s^T, \quad q(x) \sim \mathcal{N}(0, C).$$

With this $q(x)$, we define $q(y, x; \eta)$ as,

$$q(y, x; \eta) = q(x)q(y|x; \eta) = \exp\left(-\frac{1}{2}(y\ x^T)B\begin{pmatrix} y \\ x \end{pmatrix} - \psi(\eta)\right) \tag{6}$$

$$B = \frac{1}{s^2}\left(\begin{array}{c|c} 1 & -r^T \\ \hline -r & s^2 C^{-1} + rr^T \end{array}\right), \psi(\eta) = \frac{1}{2}\left(\log s^2 + \log|C| + (n+1)\log 2\pi\right).$$

## 3   THE EM AND THE *em* ALGORITHMS FOR A FACTOR ANALYSIS MODEL

It is mentioned that the W-S algorithm is similar to the EM algorithm[4]([5][7]). But there is an essential difference between them. In this section, first, we show the EM algorithm. We also describe the *em* algorithm[2] which gives us the information geometrical understanding of the EM algorithm. With these results, we will show the difference between W-S and the EM algorithms in the next section.

The EM algorithm consists of the following two steps.

**E-step:** Define $Q(\boldsymbol{\theta}, \boldsymbol{\theta}_t)$ as,

$$Q(\boldsymbol{\theta}, \boldsymbol{\theta}_t) = \frac{1}{N} \sum_{s=1}^{N} E_{p(y|\boldsymbol{x}_s; \boldsymbol{\theta}_t)} \left[ \log p(y, \boldsymbol{x}_s; \boldsymbol{\theta}) \right]$$

**M-step:** Update $\boldsymbol{\theta}$ as,

$$\boldsymbol{\theta}_{t+1} = \underset{\boldsymbol{\theta}}{\operatorname{argmax}} \, Q(\boldsymbol{\theta}, \boldsymbol{\theta}_t),$$

$$g_{t+1} = \frac{(1 + g_t^T \Sigma_t^{-1} g_t) C \Sigma_t^{-1} g_t}{g_t^T \Sigma_t^{-1} C \Sigma_t^{-1} g_t + 1 + g_t^T \Sigma_t^{-1} g_t}, \quad \Sigma_{t+1} = \operatorname{diag}\left( C - g_{t+1} \frac{g_t^T \Sigma_t^{-1} C}{1 + g_t^T \Sigma_t^{-1} g_t} \right). \tag{7}$$

$E_p[\cdot]$ denotes taking the average with the probability distribution $p$. The iteration of these two steps converges to give the MLE.

The EM algorithm only uses the generative model, but the *em* algorithm[2] also uses the recognition model. The *em* algorithm consists of the *e* and *m* steps which are defined as the *e* and *m* projections[1] between the two manifolds $M$ and $D$. The manifolds are defined as follows.

**Model manifold $M$:** $M \stackrel{\text{def}}{=} \{ p(y, \boldsymbol{x}; \boldsymbol{\theta}) | \boldsymbol{\theta} = (\boldsymbol{g}, \operatorname{diag}(\sigma_i^2)), \boldsymbol{g} \in \mathrm{R}^n, 0 < \sigma_i < \infty \}$.

**Data manifold $D$:** $D \stackrel{\text{def}}{=} \{ q(y, \boldsymbol{x}; \boldsymbol{\eta}) | \boldsymbol{\eta} = (\boldsymbol{r}, s^2), \boldsymbol{r} \in \mathrm{R}^n, 0 < s < \infty \}$, $q(\boldsymbol{x})$ include the matrix $C$ which is defined by the data, and this is called the "data manifold".

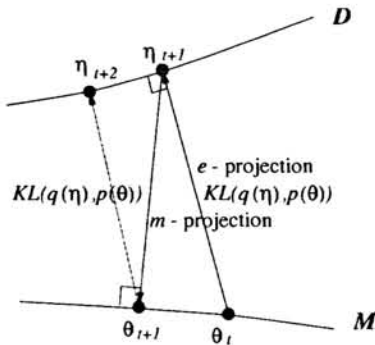

Figure 1: Information geometrical understanding of the *em* algorithm

Figure 1 schematically shows the *em* algorithm. It consists of two steps, *e* and *m* steps. On each step, parameters of recognition and generative models are updated respectively.

**e-step:** Update $\eta$ as the $e$ projection of $p(y, x; \theta_t)$ on $D$.

$$\eta_{t+1} = \underset{\eta}{\operatorname{argmin}} \, KL(q(\eta), p(\theta_t)) \tag{8}$$

$$r_{t+1} = \frac{\Sigma_t^{-1} g_t}{1 + g_t^T \Sigma_t^{-1} g_t}, \qquad s_{t+1}^2 = \frac{1}{1 + g_t^T \Sigma_t^{-1} g_t}. \tag{9}$$

where $KL(q(\eta), p(\theta))$ is the Kullback-Leibler divergence defined as,

$$KL(q(\eta), p(\theta)) = E_{q(y,x;\eta)} \left[ \log \frac{q(y, x; \eta)}{p(y, x; \theta)} \right]$$

**m-step:** Update $\theta$ as the $m$ projection of $q(y, x; \eta_t)$ on $M$.

$$\theta_{t+1} = \underset{\theta}{\operatorname{argmin}} \, KL(q(\eta_{t+1}), p(\theta)) \tag{10}$$

$$g_{t+1} = \frac{Cr_{t+1}}{s_{t+1}^2 + r_{t+1}^T C r_{t+1}}, \qquad \Sigma_{t+1} = \operatorname{diag}\left(C - g_{t+1} r_{t+1}^T C\right). \tag{11}$$

By substituting (9) for $r_{t+1}$ and $s_{t+1}^2$ in (11), it is easily proved that (11) is equivalent to (7), and the *em* and EM algorithms are equivalent.

## 4   THE DIFFERENCE BETWEEN THE W-S AND THE EM ALGORITHMS

The wake-phase corresponds to a gradient flow of the M-step[7] in the stochastic sense. But the sleep-phase is not a gradient flow of the E-step. In order to see these clear, we show the detail of the W-S phases in this section.

First, we show the averages of (1), (2), (3) and (4),

$$g_{t+1} = g_t - \alpha(s_t^2 + r_t^T C r_t)\left(g_t - \frac{Cr_t}{s_t^2 + r_t^T C r_t}\right) \tag{12}$$

$$\Sigma_{t+1} = \Sigma_t - (1 - \beta)\left(\Sigma_t - \operatorname{diag}\left(C - 2(Cr_t)g_t^T + (s_t^2 + r_t^T C r_t)g_t g_t^T\right)\right) \tag{13}$$

$$r_{t+1} = r_t - \alpha(\Sigma_{t+1} + g_{t+1} g_{t+1}^T)\left(r_t - \frac{\Sigma_{t+1}^{-1} g_{t+1}}{1 + g_{t+1}^T \Sigma_{t+1}^{-1} g_{t+1}}\right) \tag{14}$$

$$s_{t+1}^2 = s_t^2 - (1 - \beta)\left(s_t^2 - ((1 - g_{t+1}^T r_t)^2 + r_t^T \Sigma_{t+1} r_t)\right). \tag{15}$$

As the K-L divergence is rewritten as $KL(q(\eta), p(\theta))$,

$$KL(q(\eta), p(\theta)) = \frac{1}{2} tr(B^{-1}A) - \frac{n+1}{2} + \psi(\theta) - \psi(\eta),$$

the derivatives of this K-L divergence with respect to $\theta = (g, \Sigma)$ are,

$$\frac{\partial}{\partial g} KL(q(\eta), p(\theta)) = 2\left((s^2 + r^T C r)\Sigma^{-1}\right)\left(g - \frac{Cr}{s^2 + r^T C r}\right) \tag{16}$$

$$\frac{\partial}{\partial \Sigma} KL(q(\eta), p(\theta)) = \Sigma^{-2}\left(\Sigma - \operatorname{diag}\left(C - 2Crg^T + (s^2 + r^T C r)gg^T\right)\right) \tag{17}$$

With these results, we can rewrite the wake-phase as,

$$g_{t+1} = g_t - \frac{\alpha}{2}\Sigma_t \frac{\partial}{\partial g_t} KL(q(\eta_t), p(\theta_t)) \tag{18}$$

$$\Sigma_{t+1} = \Sigma_t - (1 - \beta)\Sigma_t^2 \frac{\partial}{\partial \Sigma_t} KL(q(\eta_t), p(\theta_t)) \tag{19}$$

Since $\Sigma$ is a positive definite matrix, the wake-phase is a gradient flow of $m$-step which is defined as (10).

On the other hand, $KL(p(\boldsymbol{\theta}), q(\boldsymbol{\eta}))$ is,

$$KL(p(\boldsymbol{\theta}), q(\boldsymbol{\eta})) = \frac{1}{2}tr(A^{-1}B) - \frac{n}{2} + \psi(\boldsymbol{\eta}) - \psi(\boldsymbol{\theta}).$$

The derivatives of this K-L divergence respect to $r$ and $s^2$ are,

$$\frac{\partial}{\partial r}KL(p(\boldsymbol{\theta}), q(\boldsymbol{\eta})) = \frac{2}{s^2}(\Sigma + gg^T)\left(r - \frac{\Sigma^{-1}g}{1 + g^T\Sigma^{-1}g}\right) \tag{20}$$

$$\frac{\partial}{\partial(s^2)}KL(p(\boldsymbol{\theta}), q(\boldsymbol{\eta})) = \frac{1}{(s^2)^2}\left(s^2 - ((1 - g^Tr)^2 + r^T\Sigma r)\right). \tag{21}$$

Therefore, the sleep-phase can be rewritten as,

$$r_{t+1} = r_t - \frac{\alpha}{2}s_t^2\frac{\partial}{\partial r_t}KL(p(\boldsymbol{\theta}_{t+1}), q(\boldsymbol{\eta}_t)) \tag{22}$$

$$s_{t+1}^2 = s_t^2 - (1 - \beta)(s_t^2)^2\frac{\partial}{\partial(s_t^2)}KL(p(\boldsymbol{\theta}_{t+1}), q(\boldsymbol{\eta}_t)). \tag{23}$$

These are also a gradient flow, but because of the asymmetricity of K-L divergence, (22), (23) are different from the on-line version of the $m$-step. This is the essential difference between the EM and W-S algorithms. Therefore, we cannot prove the convergence of the W-S algorithm based on the similarity of these two algorithms[7].

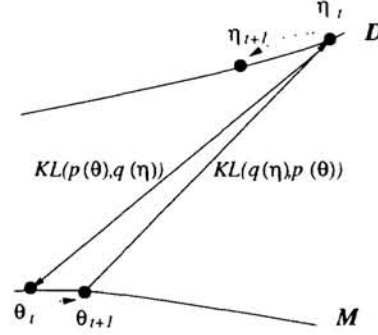

Figure 2: The Wake-Sleep algorithm

## 5 CONVERGENCE PROPERTY

We want to prove the convergence property of the W-S algorithm. If we can find a Lyapnov function for the W-S algorithm, the convergence is guaranteed[7]. But we couldn't find it. Instead of finding a Lyapnov function, we take the continuous time, and see the behavior of the parameters and K-L divergence, $KL(q(\boldsymbol{\eta}_t), p(\boldsymbol{\theta}_t))$.

$KL(q(\boldsymbol{\eta}), p(\boldsymbol{\theta}))$ is a function of $g$, $r$, $\Sigma$ and $s^2$. The derivatives with respect to $g$ and $\Sigma$ are given in (16) and (17). The derivatives with respect to $r$ and $s^2$ are,

$$\frac{\partial}{\partial r}KL(q(\boldsymbol{\eta}), p(\boldsymbol{\theta})) = 2(1 + g^T\Sigma^{-1}g)C\left(r - \frac{\Sigma^{-1}g}{1 + g^T\Sigma^{-1}g}\right) \tag{24}$$

$$\frac{\partial}{\partial(s^2)}KL(q(\boldsymbol{\eta}), p(\boldsymbol{\theta})) = 1 + g^T\Sigma^{-1}g - \frac{1}{s^2}. \tag{25}$$

On the other hand, we set the flows of $g$, $r$, $\Sigma$ and $s^2$ to follow the updating due to the W-S algorithm, that is,

$$\frac{d}{dt}g = -\alpha'(s_t^2 + r_t^T C r_t)\left(g_t - \frac{C r_t}{s_t^2 + r_t^T C r_t}\right) \tag{26}$$

$$\frac{d}{dt}r = -\alpha'(\Sigma_t + g_t g_t^T)\left(r_t - \frac{\Sigma_t^{-1} g_t}{1 + g_t^T \Sigma_t^{-1} g_t}\right) \tag{27}$$

$$\frac{d}{dt}\Sigma = -\beta'\left(\Sigma_t - \mathrm{diag}\left(C - 2C r_t g^T{}_t + (s_t^2 + r_t^T C r_t)g_t g_t^T\right)\right) \tag{28}$$

$$\frac{d}{dt}(s^2) = -\beta'\left(s_t^2 - ((1 - g_t^T r_t)^2 + r_t^T \Sigma_t r_t)\right) \tag{29}$$

With theses results, $dKL(q(\eta_t), p(\theta_t))/dt$ is,

$$\frac{dKL(q(\eta_t), p(\theta_t))}{dt} = \frac{\partial KL}{\partial g}\frac{dg}{dt} + \frac{\partial KL}{\partial r}\frac{dr}{dt} + \frac{\partial KL}{\partial \Sigma}\frac{d\Sigma}{dt} + \frac{\partial KL}{\partial (s^2)}\frac{d(s^2)}{dt}. \tag{30}$$

First 3 terms in the right side of (30) are apparently non-positive. Only the 4th one is not clear.

$$\frac{\partial KL}{\partial (s^2)}\frac{d(s^2)}{dt} = -\beta'\left(s_t^2 - ((1 - g_t^T r_t)^2 + r_t^T \Sigma_t r_t)\right)\left(1 + g_t^T \Sigma_t^{-1} g_t - \frac{1}{s_t^2}\right)$$

$$= -\frac{1 + g_t^T \Sigma_t^{-1} g_t}{s_t^2}\left(s_t^2 - ((1 - g_t^T r_t)^2 + r_t^T \Sigma_t r_t)\right)\left(s_t^2 - \frac{1}{1 + g_t^T \Sigma_t^{-1} g_t}\right).$$

The $KL(q(\eta_t), p(\theta_t))$ does not decrease when $s_t^2$ stays between $((1 - g_t^T r_t)^2 + r_t^T \Sigma_t r_t)$ and $1/(1 + g_t^T \Sigma_t^{-1} g_t)$, but if the following equation holds, these two are equivalent,

$$r_t = \frac{\Sigma_t^{-1} g_t}{1 + g_t^T \Sigma_t^{-1} g_t}. \tag{31}$$

From the above results, the flows of $g$, $r$ and $\Sigma$ decrease $KL(q(\eta_t), p(\theta_t))$ at any time. $s_t^2$ converge to $((1 - g_t^T r_t)^2 + r_t^T \Sigma_t r_t)$ but it does not always decrease $KL(q(\eta_t), p(\theta_t))$. But since $r$ does converge to satisfy (31) independently of $s_t^2$, finally $s_t^2$ converges to $1/(1 + g_t^T \Sigma_t^{-1} g_t)$.

## 6   DISCUSSION

This factor analysis model has a special property that $p(y|x; \theta)$ and $q(y|x; \eta)$ are equivalent when following conditions are satisfied[7],

$$r = \frac{\Sigma^{-1} g}{1 + g^T \Sigma^{-1} g}, \qquad s^2 = \frac{1}{1 + g^T \Sigma^{-1} g}. \tag{32}$$

From this property, minimizing $KL(p(\theta), q(\eta))$ and $KL(q(\eta), p(\theta))$ with respect to $\eta$ leads to the same point.

$$KL(p(\theta), q(\eta)) = E_{p(x;\theta)}\left[\log \frac{p(x; \theta)}{q(x)}\right] + E_{p(y,x;\theta)}\left[\log \frac{p(y|x; \theta)}{q(y|x; \eta)}\right] \tag{33}$$

$$KL(q(\eta), p(\theta)) = E_{q(x)}\left[\log \frac{q(x)}{p(x; \theta)}\right] + E_{q(y,x;\eta)}\left[\log \frac{q(y|x; \eta)}{p(y|x; \theta)}\right], \tag{34}$$

both of (33) and (34) include $\eta$ only in the second term of the right side. If (32) holds, those two terms are 0. Therefore $KL(p(\theta), q(\eta))$ and $KL(q(\eta), p(\theta))$ are minimized at the same point.

We can use this result to modify the W-S algorithm. If the factor analysis model does not try wake- and sleep- phase alternately but "sleeps well" untill convergence, it will find the $\eta$ which is equivalent to the $e$-step in the $em$ algorithm. Since the wake-phase is a gradient flow of the $m$-step, this procedure will converge to the MLE. This algorithm is equivalent to what is called the GEM(Generalized EM) algorithm[6].

The reason of the GEM and the W-S algorithms work is that $p(y|x;\theta)$ is realizable with the recognition model $q(y|x;\eta)$. If the recognition model is not realizable, the W-S algorithm won't converge to the MLE. We are going to show an example and conclude this article.

Suppose the case that the average of $y$ in the recognition model is not a linear function of $r$ and $x$ but comes through a nonlinear function $f(\cdot)$ as,

**Recognition model** $\qquad\qquad\qquad y = f(r^T x) + \delta,$

where $f(\cdot)$ is a function of single input and output and $\delta \sim \mathcal{N}(0, s^2)$ is the noise. In this case, the generative model is not realizable by the recognition model in general. And minimizing (33) with respect to $\eta$ leads to a different point from minimizing (34). $KL(p(\theta), q(\eta))$ is minimized when $r$ and $s^2$ satisfies,

$$E_{p(x;\theta)}\left[f(r^T x)f'(r^T x)x\right] = E_{p(y,x;\theta)}\left[yf'(r^T x)x\right] \tag{35}$$

$$s^2 = 1 - E_{p(y,x;\theta)}\left[-2yf(r^T x) + f^2(r^T x)\right], \tag{36}$$

while $KL(q(\eta), p(\theta))$ is minimized when $r$ and $s^2$ satisfies,

$$(1 + g^T \Sigma^{-1} g)E_{q(x;\eta)}\left[f(r^T x)f'(r^T x)x\right] = E_{q(x;\eta)}\left[f'(r^T x)xx^T\right]\Sigma^{-1}g \tag{37}$$

$$s^2 = \frac{1}{1 + g^T \Sigma^{-1} g}. \tag{38}$$

Here, $f'(\cdot)$ is the derivative of $f(\cdot)$. If $f(\cdot)$ is a linear function, $f'(\cdot)$ is a constant value and (35), (36) and (37), (38) give the same $\eta$ as (32), but these are different in general.

We studied a factor analysis model, and showed that the W-S algorithm works on this model. From further analysis, we could show that the reason why the algorithm works on the model is that the generative model is realizable by the recognition model. We also showed that the W-S algorithm doesn't converge to the MLE if the generative model is not realizable with a simple example.

## Acknowledgment

We thank Dr. Noboru Murata for very useful discussions on this work.

## References

[1] Shun-ichi Amari. *Differential-Geometrical Methods in Statistics*, volume 28 of *Lecture Notes in Statistics*. Springer-Verlag, Berlin, 1985.

[2] Shun-ichi Amari. Information geometry of the EM and em algorithms for neural networks. *Neural Networks*, 8(9):1379–1408, 1995.

[3] Peter Dayan, Geoffrey E. Hinton, and Radford M. Neal. The Helmholtz machine. *Neural Computation*, 7(5):889–904, 1995.

[4] A. P. Dempster, N. M. Laird, and D. B. Rubin. Maximum likelihood from incomplete data via the EM algorithm. *J. R. Statistical Society, Series B*, 39:1–38, 1977.

[5] G. E. Hinton, P. Dayan, B. J. Frey, and R. M. Neal. The "wake-sleep" algorithm for unsupervised neural networks. *Science*, 268:1158–1160, 1995.

[6] Geoffrey J. McLachlan and Thriyambakam Krishnan. *The EM Algorithm and Extensions*. Wiley series in probability and statistics. John Wiley & Sons, Inc., 1997.

[7] Radford M. Neal and Peter Dayan. Factor analysis using delta-rule wake-sleep learning. *Neural Computation*, 9(8):1781–1803, 1997.